# Finding the Key to a Synapse

**Thomas Natschläger & Wolfgang Maass**
Institute for Theoretical Computer Science
Technische Universität Graz, Austria
{tnatschl, maass}@igi.tu-graz.ac.at

## Abstract

Experimental data have shown that synapses are heterogeneous: different synapses respond with different sequences of amplitudes of postsynaptic responses to the same spike train. Neither the role of synaptic dynamics itself nor the role of the heterogeneity of synaptic dynamics for computations in neural circuits is well understood. We present in this article methods that make it feasible to compute for a given synapse with known synaptic parameters the spike train that is optimally fitted to the synapse, for example in the sense that it produces the largest sum of postsynaptic responses. To our surprise we find that most of these optimally fitted spike trains match common firing patterns of specific types of neurons that are discussed in the literature.

## 1 Introduction

A large number of experimental studies have shown that biological synapses have an inherent dynamics, which controls how the pattern of amplitudes of postsynaptic responses depends on the temporal pattern of the incoming spike train. Various quantitative models have been proposed involving a small number of characteristic parameters, that allow us to predict the response of a given synapse to a given spike train once proper values for these characteristic synaptic parameters have been found. The analysis of this article is based on the model of [1], where three parameters $U$, $F$, $D$ control the dynamics of a synapse and a fourth parameter $A$ – which corresponds to the synaptic "weight" in static synapse models – scales the absolute sizes of the postsynaptic responses. The resulting model predicts the amplitude $A_k$ for the $k^{th}$ spike in a spike train with interspike intervals (ISI's) $\Delta_1, \Delta_2, \ldots, \Delta_{k-1}$ through the equations[1]

$$
\begin{aligned}
A_k &= A \cdot u_k \cdot R_k \\
u_k &= U + u_{k-1}(1 - U)\exp(-\Delta_{k-1}/F) \\
R_k &= 1 + (R_{k-1} - u_{k-1}R_{k-1} - 1)\exp(-\Delta_{k-1}/D)
\end{aligned}
\tag{1}
$$

which involve two hidden dynamic variables $u \in [0, 1]$ and $R \in [0, 1]$ with the initial conditions $u_1 = U$ and $R_1 = 1$ for the first spike. These dynamic variables evolve in dependence of the synaptic parameters $U$, $F$, $D$ and the interspike intervals of the incoming

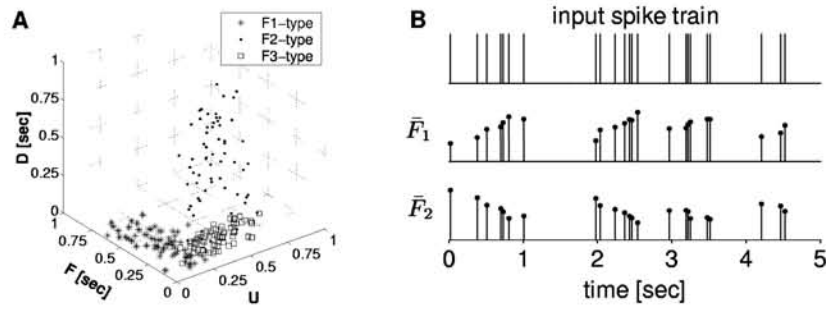

Figure 1: Synaptic heterogeneity. **A** The parameters $U$, $D$, and $F$ can be determined for biological synapses. Shown is the distribution of values for inhibitory synapses investigated in [2] which can be grouped into three mayor classes: facilitating (F1), depressing (F2) and recovering (F3). **B** Synapses produce quite different outputs for the same input for different values of the parameters $U$, $D$, and $F$. Shown are the amplitudes $u_k \cdot R_k$ (height of vertical bar) of the postsynaptic response of a F1-type and a F2-type synapse to an irregular input spike train. The parameters for synapses $\bar{F}_1$ and $\bar{F}_2$ are the mean values for the synapse types F1 and F2 reported in [2]: $\langle U, D, F \rangle = \langle 0.16, 45\,\text{msec}, 376\,\text{msec} \rangle$ for $\bar{F}_1$, and $\langle 0.25, 706\,\text{msec}, 21\,\text{msec} \rangle$ for $\bar{F}_2$.

spike train.[2] It is reported in [2] that the synaptic parameters $U$, $F$, $D$ are quite heterogeneous, even within a single neural circuit (see Fig. 1A). Note that the time constants $D$ and $F$ are in the range of a few hundred msec. The synapses investigated in [2] can be grouped into three major classes: facilitating (F1), depressing (F2) and recovering (F3). Fig. 1B compares the output of a typical F1-type and a typical F2-type synapse in response to a typical irregular spike train. One can see that the same input spike train yields markedly different outputs at these two synapses.

In this article we address the question which temporal pattern of a spike train is optimally fitted to a given synapse characterized by the three parameters $U$, $F$, $D$ in a certain sense. One possible choice is to look for the temporal pattern of a spike train which produces the largest integral of synaptic current. Note that in the case where the dendritic integration is approximately linear the integral of synaptic current is proportional to the sum $\sum_{k=1}^{N} A \cdot u_k \cdot R_k$ of postsynaptic responses. We would like to stress, that the computational methods we will present are not restricted to any particular choice of the optimality criterion. For example one can use them also to compute the spike train which produces the largest peak of the postsynaptic membrane voltage. However, in the following we will focus on the question which temporal pattern of a spike train produces the largest sum $\sum_{k=1}^{N} A \cdot u_k \cdot R_k$ of postsynaptic responses (or equivalently the largest integral of postsynaptic current).

More precisely, we fix a time interval $T$, a minimum value $\Delta_{\min}$ for ISI's, a natural number $N$, and synaptic parameters $U$, $F$, $D$. We then look for that spike train with $N$ spikes during $T$ and ISI's $\geq \Delta_{\min}$ that maximizes $\sum_{k=1}^{N} A \cdot u_k \cdot R_k$. Hence we seek for a solution — that is a sequence of ISI's $\Delta_1, \Delta_2, \ldots, \Delta_{N-1}$ — to the optimization problem

$$\text{maximize} \sum_{k=1}^{N} A \cdot u_k \cdot R_k \text{ under } \sum_{k=1}^{N-1} \Delta_k \leq T \text{ and } \Delta_{\min} \leq \Delta_k, \ 1 \leq k < N. \quad (2)$$

In Section 2 of this article we present an algorithmic approach based on dynamic program-

ming that is guaranteed to find the optimal solution of this problem (up to discretization errors), and exhibit for major types of synapses temporal patterns of spike trains that are optimally fitted to these synapses. In Section 3 we present a faster heuristic method for computing optimally fitted spike trains, and apply it to analyze how their temporal pattern depends on the number $N$ of allowed spikes during time interval $T$, i.e., on the firing rate $f = N/T$. Furthermore we analyze in Section 3 how changes in the synaptic parameters $U, F, D$ affect the temporal pattern of the optimally fitted spike train.

## 2   Computing Optimal Spike Trains for Common Types of Synapses

**Dynamic Programming**   For $T = 1000$ msec and $N = 10$ there are about $2^{100}$ spike trains among which one wants to find the optimally fitted one. We show that a computationally feasible solution to this complex optimization problem can be achieved via dynamic programming. We refer to [3] for the mathematical background of this technique, which also underlies the computation of optimal policies in reinforcement learning. We consider the discrete time dynamic system described by the equation

$$x_1 = \langle U, 1, 0 \rangle \quad \text{and} \quad x_{k+1} = g(x_k, a_k) \quad \text{for} \quad k = 1, \dots, N-1 \tag{3}$$

where $x_k$ describes the state of the system at step $k$, and $a_k$ is the "control" or "action" taken at step $k$. In our case $x_k$ is the triple $\langle u_k, R_k, t_k \rangle$ consisting of the values of the dynamic variables $u$ and $R$ used to calculate the amplitude $A \cdot u_k \cdot R_k$ of the $k^{th}$ postsynaptic response, and the time $t_k$ of the arrival of the $k^{th}$ spike at the synapse. The "action" $a_k$ is the length $\Delta_k \in [\Delta_{\min}, T - t_k]$ of the $k^{th}$ ISI in the spike train that we construct, where $\Delta_{\min}$ is the smallest possible size of an ISI (we have set $\Delta_{\min} = 5$ msec in our computations). As the function $g$ in Eq. (3) we take the function which maps $\langle u_k, R_k, t_k \rangle$ and $\Delta_k$ via Eq. (1) on $\langle u_{k+1}, R_{k+1}, t_{k+1} \rangle$ for $t_{k+1} = t_k + \Delta_k$. The "reward" for the $k^{th}$ spike is $A \cdot u_k \cdot R_k$, i.e., the amplitude of the postsynaptic response for the $k^{th}$ spike. Hence maximizing the total reward $J(x_1) = \sum_{k=1}^{N} A \cdot u_k \cdot R_k$ is equivalent to solving the maximization problem (2). The maximal possible value of $J_1(x_1)$ can be computed exactly via the equations

$$\begin{aligned} J_N(x_N) &= A \cdot u_N \cdot R_N \\ J_k(x_k) &= \max_{\Delta \in [\Delta_{\min}, T - t_k]} (A \cdot u_k \cdot R_k + J_{k+1}(g(x_k, \Delta))) \end{aligned} \tag{4}$$

backwards from $k = N - 1$ to $k = 1$. Thus the optimal sequence $a_1, \dots, a_{N-1}$ of "actions" is the sequence $\Delta_1, \dots, \Delta_{N-1}$ of ISI's that achieves the maximal possible value of $\sum_{k=1}^{N} A \cdot u_k \cdot R_k$. Note that the evaluation of $J_k(x_k)$ for a single value of $x_k$ requires the evaluation of $J_{k+1}(x_{k+1})$ for many different values of $x_{k+1}$.[3]

**The "Key" to a Synapse**   We have applied the dynamic programming approach to three major types of synapses reported in [2]. The results are summarized in Fig. 2 to Fig. 5. We refer informally to the temporal pattern of $N$ spikes that maximizes the response of a particular synapse as the "key" to this synapse. It is shown in Fig. 3 that the "keys" for the inhibitory synapses $\bar{F}_1$ and $\bar{F}_2$ are rather specific in the sense that they exhibit a substantially smaller postsynaptic response on any other of the major types of inhibitory synapses reported in [2]. The specificity of a "key" to a synapse is most pronounced for spiking frequencies $f$ below 20 Hz. One may speculate that due to this feature a neuron can activate — even without changing its firing rate — a particular subpopulation of its target neurons by generating a series of action potentials with a suitable temporal pattern, see

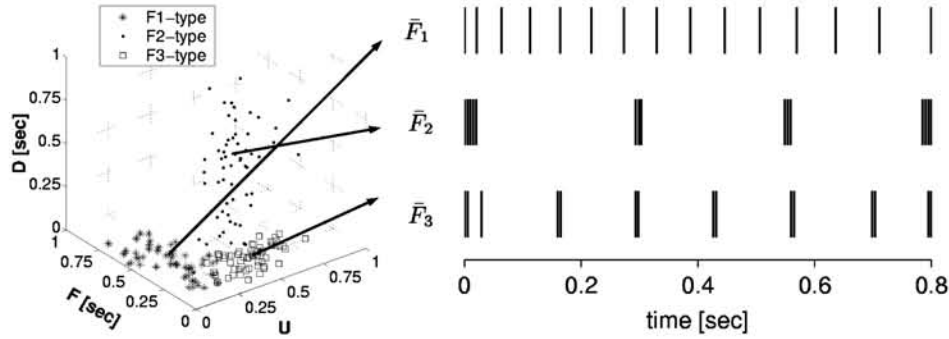

Figure 2: Spike trains that maximize the sum of postsynaptic responses for three common types of synapses ($T = 0.8\,\text{sec}$, $N = 15$ spikes). The parameters for synapses $\bar{F}_1$, $\bar{F}_2$, and $\bar{F}_3$ are the mean values for the synapse types F1, F2 and F3 reported in [2]: $\langle U, D, F \rangle = \langle 0.16, 45\,\text{msec}, 376\,\text{msec} \rangle$ for $\bar{F}_1$, $\langle 0.25, 706\,\text{msec}, 21\,\text{msec} \rangle$ for $\bar{F}_2$, and $\langle 0.32, 144\,\text{msec}, 62\,\text{msec} \rangle$ for $\bar{F}_3$.

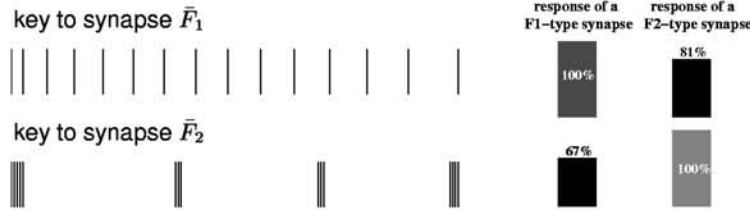

Figure 3: Specificity of optimal spike trains. The optimal spike trains for synapses $\bar{F}_1$ and $\bar{F}_2$ — the "keys" to the synapses $\bar{F}_1$ and $\bar{F}_2$ — obtained for $T = 0.8\,\text{sec}$ and $N = 15$ spikes are tested on the synapses $\bar{F}_1$ and $\bar{F}_2$. If the "key" to synapse $\bar{F}_1$ ($\bar{F}_2$) is tested on the synapse $\bar{F}_1$ ($\bar{F}_2$) this synapse produces the maximal (100 %) postsynaptic response. If on the other hand the "key" to synapse $\bar{F}_1$ ($\bar{F}_2$) is tested on synapse $\bar{F}_2$ ($\bar{F}_1$) this synapse produces significantly less postsynaptic response.

Fig. 4. Recent experiments [5, 6] show that neuromodulators can control the firing mode of cortical neurons. In [5] it is shown that bursting neurons may switch to regular firing if norepinephrine is applied. Together with the specificity of synapses to certain temporal patterns these findings point to one possible mechanism how neuromodulators can change the effective connectivity of a neural circuit.

**Relation to discharge patterns** A noteworthy aspect of the "keys" shown in Fig. 2 (and in Fig. 6 and Fig. 7) is that they correspond to common firing patterns ("accommodating", "non-accommodating", "stuttering", "bursting" and "regular firing") of neocortical interneurons reported under controlled conditions in vitro [2, 5] and in vivo [7]. For example the temporal patterns of the "keys" to the synapses $\bar{F}_1$, $\bar{F}_2$, and $\bar{F}_3$ are similar to the discharge patterns of "accommodating" [2], "bursting" [5, 7], and "stuttering" [2] cells respectively.

**What is the role of the parameter $A$?** Another interesting effect arises if one compares the optimal values of the sum $\sum_{k=1}^{N} u_k \cdot R_k$ (i.e. $A = 1$) for synapses $\bar{F}_1$, $\bar{F}_2$, and $\bar{F}_3$ (see Fig. 5A) with the maximal values of $\sum_{k=1}^{N} A \cdot u_k \cdot R_k$ (see Fig. 5B), where we have set

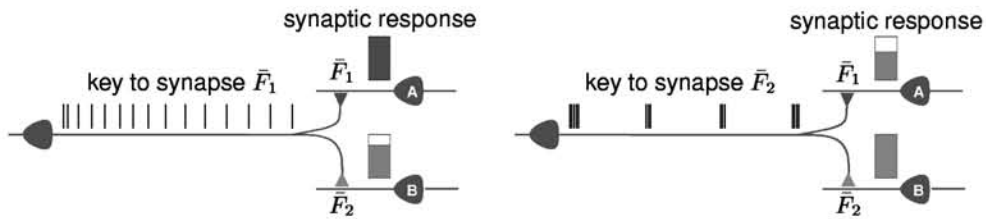

Figure 4: Preferential addressing of postsynaptic targets. Due to the specificity of a "key" to a synapse a presynaptic neuron may address (i.e. evoke stronger response at) either neuron A or B, depending on the temporal pattern of the spike train (with the same frequency $f = N/T$) it produces ($T = 0.8$ sec and $N = 15$ in this example).

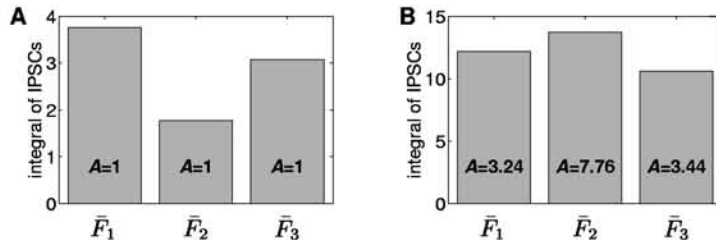

Figure 5: **A** Absolute values of the sums $\sum_{k=1}^{N} u_k \cdot R_k$ if the key to synapse $\bar{F}_i$ is applied to synapse $\bar{F}_i$, $i = 1, 2, 3$. **B** Same as panel A except that the value of $\sum_{k=1}^{N} A \cdot u_k \cdot R_k$ is plotted. For $A$ we used the value of $G_{\max}$ (in nS) reported in [2]. The quotient max / min is 1.3 compared to 2.13 in panel A.

$A$ equal to the value of $G_{\max}$ reported in [2]. Whereas the values of $G_{\max}$ vary strongly among different synapse types (see Fig. 5B), the resulting maximal response of a synapse to its proper "key" is almost the same for each synapse. Hence, one may speculate that the system is designed in such a way that each synapse should have an equal influence on the postsynaptic neuron when it receives its optimal spike train. However, this effect is most evident for a spiking frequency $f = N/T$ of 10 Hz and vanishes for higher frequencies.

## 3 Exploring the Parameter Space

**Sequential Quadratic Programming**   The numerical approach for approximately computing optimal spike trains that was used in section 2 is sufficiently fast so that an average PC can carry out any of the computations whose results were reported in Fig. 2 within a few hours. To be able to address computationally more expensive issues we used a a nonlinear optimization algorithm known as "sequential quadratic programming" (SQP)[4] which is the state of the art approach for heuristically solving constrained optimization problems such as (2). We refer the reader to [8] for the mathematical background of this technique and to [4] for more details about the application of SQP for approximately computing optimal spike trains.

**Optimal Spike Trains for Different Firing Rates**   First we used SQP to explore the effect of the spike frequency $f = N/T$ on the temporal pattern of the optimal spike train. For the synapses $\bar{F}_1$, $\bar{F}_2$, and $\bar{F}_3$ we computed the optimal spike trains for frequencies

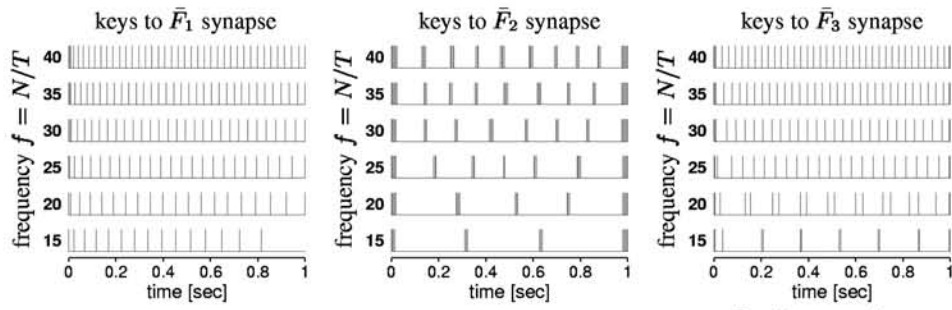

Figure 6: Dependence of the optimal spike train of the synapses $\bar{F}_1$, $\bar{F}_2$, and $\bar{F}_3$ on the spike frequency $f = N/T$ ($T = 1\,\text{sec}$, $N = 15, \ldots, 40$).

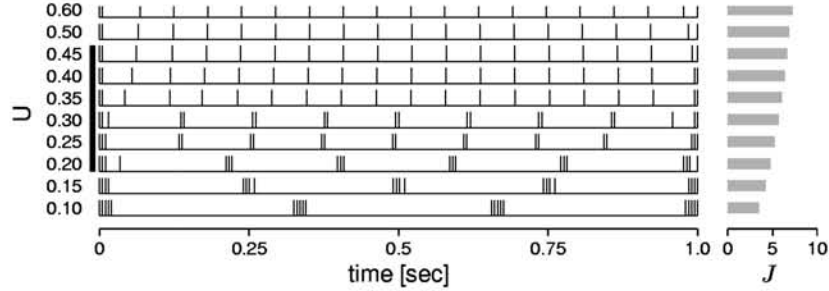

Figure 7: Dependence of the optimal spike train on the synaptic parameter $U$. It is shown how the optimal spike train changes if the parameter $U$ is varied. The other two parameters are set to the value corresponding to synapse $\bar{F}_3$: $D = 144\,\text{msec}$ and $F = 62\,\text{msec}$. The black bar to the left marks the range of values (mean $\pm$ std) reported in [2] for the parameter $U$. To the right of each spike train we have plotted the corresponding value of $J = \sum_{k=1}^{N} u_k R_k$ (gray bars).

ranging from $15\,\text{Hz}$ to $40\,\text{Hz}$. The results are summarized in Fig. 6. For synapses $\bar{F}_1$ and $\bar{F}_2$ the characteristic spike pattern ($\bar{F}_1$ ... accommodating, $\bar{F}_2$ ... stuttering) is the same for all frequencies. In contrast, the optimal spike train for synapse $\bar{F}_3$ has a phase transition from "stuttering" to "non-accommodating" at about $20\,\text{Hz}$.

**The Impact of Individual Synaptic Parameters**    We will now address the question how the optimal spike train depends on the individual synaptic parameters $U$, $F$, and $D$. The results for the case of F3-type synapses and the parameter $U$ are summarized in Fig. 7. For results with regard to other parameters and synapse types we refer to [4]. We have marked in Fig. 7 with a black bar the range of $U$ for F3-type synapses reported in [2]. It can be seen that within this parameter range we find "regular" and "bursting" spike patterns. Note that the sum of postsynaptic responses $J$ (gray horizontal bars in Fig. 7) is not proportional to $U$. While $U$ increases from 0.1 to 0.6 (6 fold change) $J$ only increases by a factor of 2. This seems to be interesting since the parameter $U$ is closely related to the initial release probability of a synapse, and it is a common assumption that the "strength" of a synapse is proportional to its initial release probability.

# 4 Discussion

We have presented two complementary computational approaches for computing spike trains that optimize a given response criterion for a given synapse. One of these methods is based on dynamic programming (similar as in reinforcement learning), the other one on sequential quadratic programming. These computational methods are not restricted to any particular choice of the optimality criterion and the synaptic model. In [4] applications of these methods to other optimality criteria, e.g. maximizing the specificity, are discussed.

It turns out that the spike trains that maximize the response of F1-, F2- and F3-type synapses (see Fig. 1) are well known firing patterns like "accommodating", "bursting" and "regular firing" of specific neuron types. Furthermore for F1- and F3-type synapses the optimal spike train agrees with the most often found firing pattern of presynaptic neurons reported in [2], whereas for F2-type synapses there is no such agreement; see [4]. This observation provides the first glimpse at a possible functional role of the specific combinations of synapse types and neuron types that was recently found in [2].

Another noteworthy aspect of the optimal spike trains is their specificity for a given synapse (see Fig. 3).: suitable *temporal firing patterns* activate preferentially *specific* types of synapses. One potential functional role of such specificity to temporal firing patterns is the possibility of preferential addressing of postsynaptic target neurons (see Fig. 4). Note that there is experimental evidence that cortical neurons can switch their intrinsic firing behavior from "bursting" to "regular" depending on neuromodulator mediated inputs [5, 6]. This findings provide support for the idea of preferential addressing of postsynaptic targets implemented by the interplay of dynamic synapses and the intrinsic firing behavior of the presynaptic neuron.

Furthermore our analysis provides the platform for a deeper understanding of the specific role of different synaptic parameters, because with the help of the computational techniques that we have introduced one can now see directly how the temporal structure of the optimal spike train for a synapse depends on the individual synaptic parameters. We believe that this inverse analysis is essential for understanding the computational role of neural circuits.

## Footnotes

[1]To be precise: the term $u_{k-1}R_{k-1}$ in Eq. (1) was erroneously replaced by $u_k R_{k-1}$ in the corresponding Eq. (2) of [1]. The model that they actually fitted to their data is the model considered in this article.

[2]It should be noted that this deterministic model predicts the cumulative response of a *population* of stochastic release sites that make up a synaptic connection.

[3]When one solves Eq. (4) on a computer, one has to replace the continuous state variable $x_k$ by a discrete variable $\tilde{x}_k$, and round $x_{k+1} := g(\tilde{x}_k, \Delta)$ to the nearest value of the corresponding discrete variable $\tilde{x}_{k+1}$. For more details about the discretization of the model we refer the reader to [4].

[4]We used the implementation (function `constr`) which is contained in the *MATLAB Optimization Toolbox* (see `http://www.mathworks.com/products/optimization/`).

# References

[1] H. Markram, Y. Wang, and M. Tsodyks. Differential signaling via the same axon of neocortical pyramidal neurons. *Proc. Natl. Acad. Sci.*, 95:5323–5328, 1998.

[2] A. Gupta, Y. Wang, and H. Markram. Organizing principles for a diversity of GABAergic interneurons and synapses in the neocortex. *Science*, 287:273–278, 2000.

[3] D. P. Bertsekas. *Dynamic Programming and Optimal Control, Volume 1*. Athena Scientific, Belmont, Massachusetts, 1995.

[4] T. Natschläger and W. Maass. Computing the optimally fitted spike train for a synapse. submitted for publication, electronically available via http://www.igi.TUGraz.at/igi/tnatschl/psfiles/synkey-journal.ps.gz, 2000.

[5] Z. Wang and D. A. McCormick. Control of firing mode of corticotectal and corticopontine layer V burst generating neurons by norepinephrine. *Journal of Neuroscience*, 13(5):2199–2216, 1993.

[6] J. C. Brumberg, L. G. Nowak, and D. A. McCormick. Ionic mechanisms underlying repetitive high frequency burst firing in supragranular cortical neurons. *Journal of Neuroscience*, 20(1):4829–4843, 2000.

[7] M. Steriade, I. Timofeev, N. Dürmüller, and F. Grenier. Dynamic properties of corticothalamic neurons and local cortical interneurons generating fast rhytmic (30–40 hz) spike bursts. *Journal of Neurophysiology*, 79:483–490, 1998.

[8] M. J. D. Powell. Variable metric methods for constrained optimization. In A. Bachem, M Grotschel, and B. Korte, editors, *Mathematical Programming: The State of the Art*, pages 288–311. Springer Verlag, 1983.
